# Reducing statistical dependencies in natural signals using radial Gaussianization

**Siwei Lyu**
Computer Science Department
University at Albany, SUNY
Albany, NY 12222
lsw@cs.albany.edu

**Eero P. Simoncelli**
Center for Neural Science
New York University
New York, NY 10003
eero@cns.nyu.edu

## Abstract

We consider the problem of transforming a signal to a representation in which the components are statistically independent. When the signal is generated as a linear transformation of independent Gaussian or non-Gaussian sources, the solution may be computed using a linear transformation (PCA or ICA, respectively). Here, we consider a complementary case, in which the source is non-Gaussian but elliptically symmetric. Such a source cannot be decomposed into independent components using a linear transform, but we show that a simple nonlinear transformation, which we call radial Gaussianization (RG), is able to remove all dependencies. We apply this methodology to natural signals, demonstrating that the joint distributions of nearby bandpass filter responses, for both sounds and images, are closer to being elliptically symmetric than linearly transformed factorial sources. Consistent with this, we demonstrate that the reduction in dependency achieved by applying RG to either pairs or blocks of bandpass filter responses is significantly greater than that achieved by PCA or ICA.

## 1 Introduction

Signals may be manipulated, transmitted or stored more efficiently if they are transformed to a representation in which there is no statistical redundancy between the individual components. In the context of biological sensory systems, the *efficient coding hypothesis* [1, 2] proposes that the principle of reducing redundancies in natural signals can be used to explain various properties of biological perceptual systems. Given a source model, the problem of deriving an appropriate transformation to remove statistical dependencies, based on the statistics of observed samples, has been studied for more than a century. The most well-known example is principal components analysis (PCA), a linear transformation derived from the second-order signal statistics (i.e., the covariance structure), that can fully eliminate dependencies for Gaussian sources. Over the past two decades, a more general method, known as independent component analysis (ICA), has been developed to handle the case when the signal is sampled from a linearly transformed factorial source. ICA and related methods have shown success in many applications, especially in deriving optimal representations for natural signals [3, 4, 5, 6].

Although PCA and ICA bases may be computed for nearly any source, they are only guaranteed to eliminate dependencies when the assumed source model is correct. And even in cases where these methodologies seems to produce an interesting solution, the components of the resulting representation may be far from independent. A case in point is that of natural images, for which derived ICA transformations consist of localized oriented basis functions that appear similar to the receptive field descriptions of neurons in mammalian visual cortex [3, 5, 4]. Although dependency between the responses of such linear basis functions is reduced compared to that of the original pixels, this reduc-

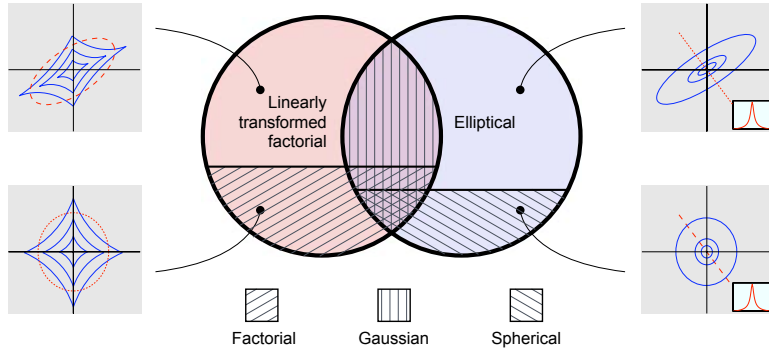

**Fig. 1.** Venn diagram of the relationship between density models. The two circles represent the linearly transformed factorial densities as assumed by the ICA methods, and elliptically symmetric densities (ESDs). The intersection of these two classes is the set of all Gaussian densities. The factorial densities form a subset of the linearly transformed factorial densities and the spherically symmetric densities form a subset of the ESDs.

tion is only slightly more than that achieved with PCA or other bandpass filters [7, 8]. Furthermore, the responses of ICA and related filters still exhibit striking higher-order dependencies [9, 10, 11].

Here, we consider the dependency elimination problem for the class of source models known as elliptically symmetric densities (ESDs) [12]. For ESDs, linear transforms have no effect on the dependencies beyond second-order, and thus ICA decompositions offer no advantage over PCA. We introduce an alternative nonlinear procedure, which we call *radial Gaussianization* (RG). In RG, the norms of whitened signal vectors are nonlinearly adjusted to ensure that the resulting output density is a spherical Gaussian, whose components are statistically independent. We first show that the joint statistics of proximal bandpass filter responses for natural signals (sounds and images) are better described as an ESD than linearly transformed factorial sources. Consistent with this, we demonstrate that the reduction in dependency achieved by applying RG to such data is significantly greater than that achieved by PCA or ICA. A preliminary version of portions of this work was described in [13].

## 2   Elliptically Symmetric Densities

The density of a random vector $\mathbf{x} \in \mathcal{R}^d$ with zero mean is elliptically symmetric if it is of the form:

$$p(\mathbf{x}) = \frac{1}{\alpha|\Sigma|^{\frac{1}{2}}} f\left(-\frac{1}{2}\mathbf{x}^T\Sigma^{-1}\mathbf{x}\right), \tag{1}$$

where $\Sigma$ is a positive definite matrix, $f(\cdot)$ is the generating function satisfying $f(\cdot) \geq 0$ and $\int_0^\infty f(-r^2/2)\, r^{d-1}\, dr < \infty$, and the normalizing constant $\alpha$ is chosen so that the density integrates to one [12]. The definitive characteristic of an ESD is that the level sets of constant probability are ellipsoids determined by $\Sigma$. In the special case when $\Sigma$ is a multiple of the identity matrix, the level sets of $p(\mathbf{x})$ are hyper-spheres and the density is known as a *spherically symmetric density* (SSD). Assuming $\mathbf{x}$ has finite second-order statistics, $\Sigma$ is a multiple of the covariance matrix, which implies that any ESD can be transformed into an SSD by a PCA/whitening operation.

When the generating function is an exponential, the resulting ESD is a zero-mean multivariate Gaussian with covariance matrix $\Sigma$. In this case, $\mathbf{x}$ can also be regarded as a linear transformation of a vector $\mathbf{s}$ containing independent unit-variance Gaussian components, as: $\mathbf{x} = \Sigma^{-1/2}\mathbf{s}$. In fact, the Gaussian is the only density that is both elliptically symmetric and linearly decomposable into independent components [14]. In other words, the Gaussian densities correspond to the intersection of the class of ESDs and the class assumed by the ICA methods. As a special case, a spherical Gaussian is the *only* spherically symmetric density that is also factorial (i.e., has independent components). These relationships are illustrated in a Venn diagram in Fig. 1.

Apart from the special case of Gaussian densities, a linear transformation such as PCA or ICA cannot completely eliminate dependencies in the ESDs. In particular, PCA and whitening can transform an ESD variable to a spherically symmetric variable, $\mathbf{x}_{\text{wht}}$, but the resulting density will not be factorial unless it is Gaussian. And ICA would apply an additional rotation (i.e., an orthogonal

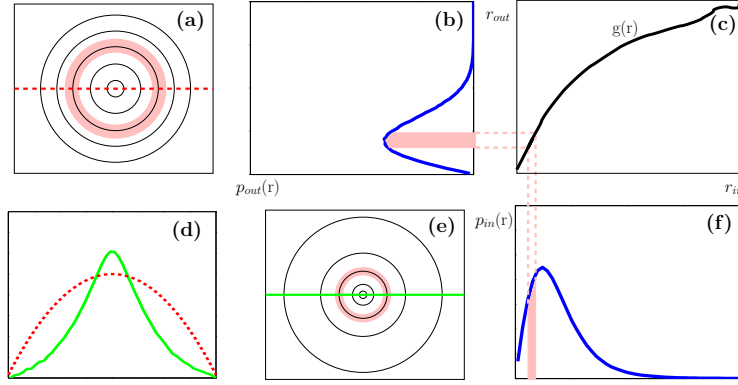

**Fig. 2.** Radial Gaussianization procedure for 2D data. **(a,e)**: 2D joint densities of a spherical Gaussian and a non-Gaussian SSD, respectively. The plots are arranged such that a spherical Gaussian has equal-spaced contours. **(b,f)**: radial marginal densities of the spherical Gaussian in (a) and the SSD in (e), respectively. Shaded regions correspond to shaded annuli in (a) and (e). **(c)**: the nonlinear mapping that transforms the radii of the source to those of the spherical Gaussian. **(d)**: log marginal densities of the Gaussian in (a) and the SSD in (e), as red dashed line and green solid line, respectively.

matrix) to transform $\mathbf{x}_{\text{wht}}$ to a new set of coordinates maximizing a higher-order contrast function (e.g., kurtosis). However, for spherically symmetric $\mathbf{x}_{\text{wht}}$, $p(\mathbf{x}_{\text{wht}})$ is invariant to rotation, and thus unaffected by orthogonal transformations.

## 3 Radial Gaussianization

Given that linear transforms are ineffective in removing dependencies from a spherically symmetric variable $\mathbf{x}_{\text{wht}}$ (and hence the original ESD variable $\mathbf{x}$), we need to consider non-linear mappings. As described previously, a spherical Gaussian is the only SSD with independent components. Thus, a natural solution for eliminating the dependencies in a non-Gaussian spherically symmetric $\mathbf{x}_{\text{wht}}$ is to transform it to a spherical Gaussian.

Selecting such a non-linear mapping without any further constraint is a highly ill-posed problem. It is natural to restrict to nonlinear mappings that act *radially*, preserving the spherical symmetry. Specifically, one can show that the generating function of $p(\mathbf{x}_{\text{wht}})$ is completely determined by its radial marginal distribution: $p_r(r) = \frac{r^{d-1}}{\beta} f(-r^2/2)$, where $r = \|\mathbf{x}_{\text{wht}}\|$, $\Gamma(\cdot)$ is the standard Gamma function, and $\beta$ is the normalizing constant that ensures that the density integrates to one. In the special case of a spherical Gaussian of unit variance, the radial marginal is a *chi*-density with $d$ degrees of freedom: $p_\chi(r) = \frac{r^{d-1}}{2^{d/2-1}\Gamma(d/2)} \exp(-r^2/2)$. We define the *radial Gaussianization* (RG) transformation as $\mathbf{x}_{\text{rg}} = g(\|\mathbf{x}_{\text{wht}}\|) \frac{\mathbf{x}_{\text{wht}}}{\|\mathbf{x}_{\text{wht}}\|}$, where nonlinear function $g(\cdot)$ is selected to map the radial marginal density of $\mathbf{x}_{\text{wht}}$ to the *chi*-density. Solving for a monotonic $g(\cdot)$ is a standard one-dimensional density-mapping problem, and the unique solution is the composition of the inverse cumulative density function (CDF) of $p_\chi$ with the CDF of $p_r$: $g(r) = F_\chi^{-1} F_r(r)$. A illustration of the procedure is provided in Fig. 2. In practice, we can estimate $F_r(r)$ from a histogram computed from training data, and use this to construct a numerical approximation (i.e., a look-up table) of the continuous function $\hat{g}(r)$. Note that the accuracy of the estimated RG transformation will depend on the number of data samples, but is independent of the dimensionality of the data vectors.

In summary, a non-Gaussian ESD signal can be radially Gaussianized by first applying PCA and whitening operations to remove second-order dependency (yielding an SSD), followed by a nonlinear transformation that maps the radial marginal to a *chi*-density.

## 4 Application to Natural Signals

An understanding of the statistical behaviors of source signals is beneficial for many problems in signal processing, and can also provide insights into the design and functionality of biological sensory systems. Gaussian signal models are widely used, because they are easily characterized and often lead to clean and efficient solutions. But many naturally occurring signals exhibit striking

non-Gaussian statistics, and much recent literature focuses on the problem of characterizing and exploiting these behaviors. Specifically, ICA methodologies have been used to derive linear representations for natural sound and image signals whose coefficients are maximally sparse or independent [3, 5, 6]. These analyses generally produced basis sets containing bandpass filters resembling those used to model the early transformations of biological auditory and visual systems.

Despite the success of ICA methods in providing a fundamental motivation for sensory receptive fields, there are a number of simple observations that indicate inconsistencies in this interpretation. First, the responses of ICA or other bandpass filters exhibit striking dependencies, in which the variance of one filter response can be predicted from the amplitude of another nearby filter response [10, 15]. This suggests that although the marginal density of the bandpass filter responses are heavy-tailed, their joint density is not consistent with the linearly transformed factorial source model assumed by ICA. Furthermore, the marginal distributions of a wide variety of bandpass filters (even a "filter" with randomly selected zero-mean weights) are *all* highly kurtotic [7]. This would not be expected for the ICA source model: projecting the local data onto a random direction should result in a density that becomes more Gaussian as the neighborhood size increases, in accordance with a generalized version of the central limit theorem [16]. A recent quantitative study [8] further showed that the oriented bandpass filters obtained through ICA optimization on images lead to a surprisingly small improvement in reducing dependency relative to decorrelation methods such as PCA. Taken together, all of these observations suggest that the filters obtained through ICA optimization represent a "shallow" optimum, and are perhaps not as uniquely suited for image or sound representation as initially believed. Consistent with this, recently developed models for local image statistics model local groups of image bandpass filter responses with non-Gaussian ESDs [e.g., 17, 18, 11, 19, 20]. These all suggest that RG might provide an appropriate means of eliminating dependencies in natural signals. Below, we test this empirically.

## 4.1 Dependency Reduction in Natural Sounds

We first apply RG to natural sounds. We used sound clips from commercial CDs, which have a sampling frequency of 44100 Hz and typical length of $15 - 20$ seconds, and contents including animal vocalization and recordings in natural environments. These sound clips were filtered with a bandpass gammatone filter, which are commonly used to model the peripheral auditory system [21]. In our experiments, analysis was based on a filter with center frequency of 3078 Hz.

Shown in the top row of column (a) in Fig.3 are contour plots of the joint histograms obtained from pairs of coefficients of a bandpass-filtered natural sound, separated with different time intervals. Similar to the empirical observations for natural images [17, 11], the joint densities are non-Gaussian, and have roughly elliptically symmetric contours for temporally proximal pairs. Shown in the top row of column (b) in Fig.3 are the conditional histograms corresponding to the same pair of signals. The "bow-tie" shaped conditional distribution, which has been also observed in natural images [10, 11, 15], indicates that the conditional variance of one signal depends on the value of the other. This is a highly non-Gaussian behavior, since the conditional variances of a jointly Gaussian density are always constant, independent of the value of the conditioning variable. For pairs that are distant, both the second-order correlation and the higher-order dependency become weaker. As a result, the corresponding joint histograms show more resemblance to the factorial product of two one-dimensional super-Gaussian densities (bottom row of column (a) in Fig.3), and the shape of the corresponding conditional histograms (column (b)) is more constant, all as would be expected for two independent random variables .

As described in previous sections, the statistical dependencies in an elliptically symmetric random variable can be effectively removed by a linear whitening operation followed by a nonlinear radial Gaussianization, the latter being implemented as histogram transform of the radial marginal density of the whitened signal. Shown in columns (c) and (d) in Fig.3 are the joint and conditional histograms of the transformed data. First, note that when the two signals are nearby, RG is highly effective, as suggested by the roughly Gaussian joint density (equally spaced circular contours), and by the consistent vertical cross-sections of the conditional histogram. However, as the temporal separation between the two signals increases, the effects of RG become weaker (middle row, Fig. 3). When the two signals are distant (bottom row, Fig.3), they are nearly independent, and applying RG can actually *increase* dependency, as suggested by the irregular shape of the conditional densities (bottom row, column (d)).

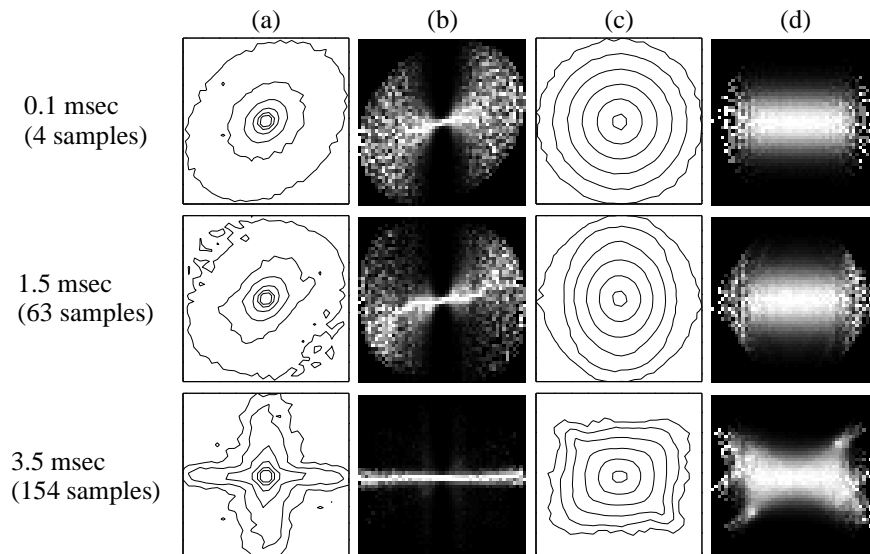

**Fig. 3.** Radial Gaussianization of natural sounds. **(a)**: Contour plots of joint histograms of pairs of band-pass filter responses of a natural sound clip. Each row corresponds to pairs with different temporal separation, and levels are chosen so that a spherical Gaussian density will have equally spaced contours. **(c)** Joint histograms after whitening and RG transformation. **(b,d)**: Conditional histograms of the same data shown in (a,c), computed by independently normalizing each column of the joint histogram. Histogram intensities are proportional to probability, except that each column of pixels is independently rescaled so that the largest probability value is displayed as white.

To quantify more precisely the dependency reduction achieved by RG, we measure the statistical dependency of our multivariate sources using the *multi-information* (MI) [22], which is defined as the Kulback-Leibler divergence [23] between the joint distribution and the product of its marginals: $I(\mathbf{x}) = D_{\mathrm{KL}}\left(p(\mathbf{x}) \| \prod_k p(x_k)\right) = \sum_{k=1}^{d} H(x_k) - H(\mathbf{x})$, where $H(\mathbf{x}) = \int p(\mathbf{x}) \, log\,(p(\mathbf{x})) \, d\mathbf{x}$ is the differential entropy of $\mathbf{x}$, and $H(x_k)$ denotes the differential entropy of the $k$th component of $\mathbf{x}$. As a measure of statistical dependency among the elements of $\mathbf{x}$, MI is non-negative, and is zero if and only if the components of $\mathbf{x}$ are mutually independent. Furthermore, MI is invariant to any transformation on individual components of $\mathbf{x}$ (e.g., element-wise rescaling).

To compare the effect of different dependency reduction methods, we estimated the MI of pairs of bandpass filter responses with different temporal separations. This is achieved with a non-parametric "bin-less" method based on the order statistics [24], which alleviates the strong bias and variance intrinsic to the more traditional binning (i.e., "plug-in") estimators. It is especially effective in this case, where the data dimensionality is two. We computed the MI for each pair of raw signals, as well as pairs of the PCA, ICA and RG transformed signals. The ICA transformation was obtained using RADICAL [25], an algorithm that directly optimizes the MI using a smoothed grid search over a non-parametric estimate of entropy.

The results, averaged over all 10 sounds, are plotted in Fig. 4. First, we note that PCA produces a relatively modest reduction in MI: roughly 20% for small separations, decreasing gradually as the separation increase. We also see that ICA offers very little additional reduction over PCA for small separations. In contrast, the nonlinear RG transformation achieves an impressive reduction (nearly 100%) in MI for pairs separated by less than 0.5 msec. This can be understood by considering the joint and conditional histograms in Fig. 3. Since the joint density of nearby pairs is approximately elliptically symmetric, ICA cannot provide much improvement beyond what is obtained with PCA, while RG is expected to perform well. On the other hand, the joint densities of more distant pairs (beyond 2.5 msec) are roughly factorial, as seen in the bottom row of Fig. 3. In this case, neither PCA nor ICA is effective in further reducing dependency, as is seen in the plots of Fig. 4, but RG makes the pairs *more* dependent, as indicated by an increase in MI above that of the original pairs for separation over 2.5 msec. This is a direct result of the fact that the data do not adhere to the elliptically symmetric source model assumptions underlying the RG procedure. For intermediate separations (0.2 to 2 msec), there is a transition of the joint densities from elliptically symmetric to factorial (second row in Fig. 3), and ICA is seen to offer a modest improvement over PCA. We

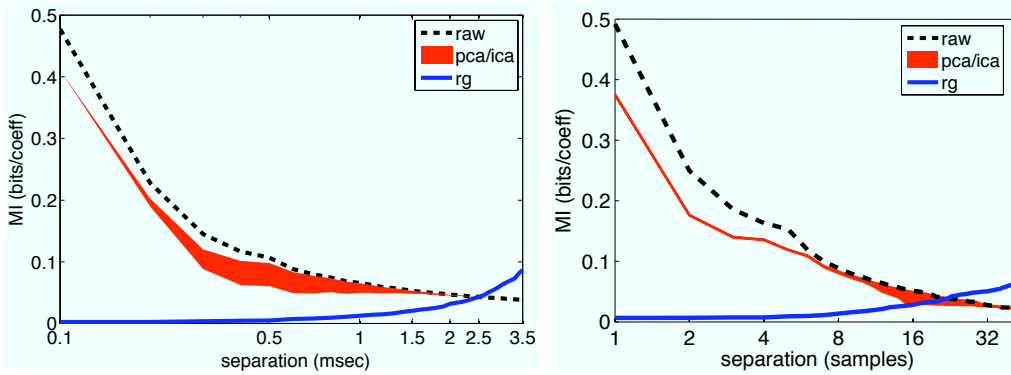

**Fig. 4. Left:** Multi-information (in bits/coefficient) for pairs of bandpass filter responses of natural audio signals, as a function of temporal separation. Shown are the MI of the raw filter response pairs, as well as the MI of the pairs transformed with PCA, ICA, and RG. Results are averaged over 10 natural sound signals. **Right:** Same analysis for pairs of bandpass filter responses averaged over 8 natural images.

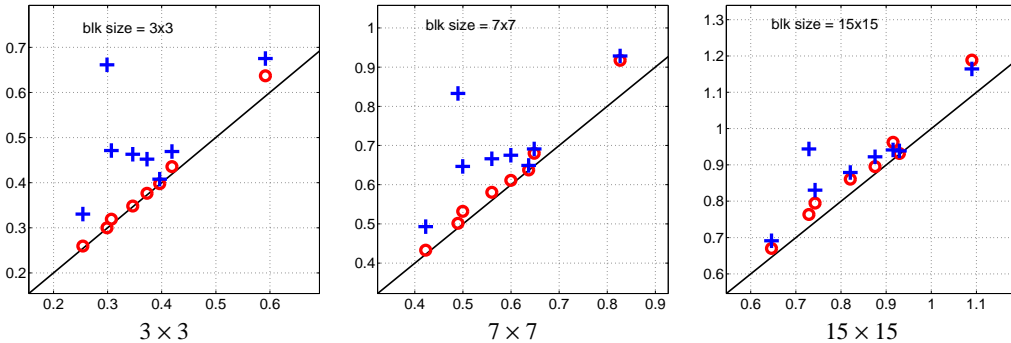

**Fig. 5.** Reduction of MI (bits/pixel) achieved with ICA and RG transforms, compared to that achieved with PCA, for pixel blocks of various sizes. The x-axis corresponds to $\Delta I_{pca}$. Pluses denotes $\Delta I_{rg}$, and circles denotes $\Delta I_{ica}$. Each plotted symbol corresponds to the result from one image in our test set.

found qualitatively similar behaviors (right column in Fig. 4) when analyzing pairs of bandpass filter responses of natural images using the data sets described in the next section.

## 4.2 Dependency Reduction in Natural Images

We have also examined the ability of RG to reduce dependencies of image pixel blocks with local mean removed. We examined eight images of natural woodland scenes from the van Hateren database [26]. We extracted the central $1024 \times 1024$ region from each, computed the log of the intensity values, and then subtracted the local mean [8] by convolving with an isotropic bandpass filter that captures an annulus of frequencies in the Fourier domain ranging from $\pi/4$ to $\pi$ radians/pixel. We denote blocks taken from these bandpass filtered images as $\mathbf{x}_{\text{raw}}$. These blocks were then transformed with PCA (denoted $\mathbf{x}_{\text{pca}}$), ICA (denoted $\mathbf{x}_{\text{ica}}$) and RG (denoted $\mathbf{x}_{\text{rg}}$). These block data are of significantly higher dimension than the filter response pairs examined in the previous section. For this reason, we switched our ICA computations from RADICAL to the more efficient FastICA algorithm [27], with a contrast function $g(u) = 1 - \exp(-u^2)$ and using the symmetric approach for optimization.

We would like to compare the dependency reduction performance of each of these methods using multi-information. However, direct estimation of MI becomes difficult and less accurate with higher data dimensionality. Instead, as in [8], we can avoid direct estimation of MI by evaluating and comparing the *differences* in MI of the various transformed blocks relative to $\mathbf{x}_{\text{raw}}$. Specifically, we use $\Delta I_{pca} = I(\mathbf{x}_{raw}) - I(\mathbf{x}_{pca})$ as a reference value, and compare this with $\Delta I_{ica} = I(\mathbf{x}_{raw}) - I(\mathbf{x}_{ica})$ and $\Delta I_{rg} = I(\mathbf{x}_{raw}) - I(\mathbf{x}_{rg})$. Full details of this computation are described in [13].

Shown in Fig.5 are scatter plots of $\Delta I_{pca}$ versus $\Delta I_{ica}$ (red circles) and $\Delta I_{rg}$ (blue pluses) for various block sizes. Each point corresponds to MI computation over blocks from one of eight bandpass-filtered test images. As the figure shows, RG achieves significant reduction in MI for most images, and this holds over a range of block sizes, whereas ICA shows only a very small improvement relative to PCA[1]. We again conclude that ICA does not offer much advantage over second-order decorrelation algorithms such as PCA, while RG offers significant improvements. These results may be attributed to the fact that the joint density for local pixel blocks tend to be close to be elliptically symmetric [17, 11].

## 5  Conclusion

We have introduced a new signal transformation known as radial Gaussianization (RG), which can eliminate dependencies of sources with elliptically symmetric densities. Empirically, we have shown that RG transform is highly effective at removing dependencies between pairs of samples in band-pass filtered sounds and images, and within local blocks of bandpass filtered images.

One important issue underlying our development of this methodology is the intimate relation between source models and dependency reduction methods. The class of elliptically symmetric densities represents a generalization of the Gaussian family that is complementary to the class of linearly transformed factorial densities (see Fig. 1). The three dependency reduction methods we have discussed (PCA, ICA and RG) are each associated with one of these classes, and are each guaranteed to produce independent responses when applied to signals drawn from a density belonging to the corresponding class. But applying one of these methods to a signal with an incompatible source model may not achieve the expected reduction in dependency (e.g., applying ICA to an ESD), and in some cases can even increase dependencies (e.g., applying RG to a factorial density).

Several recently published methods are related to RG. An iterative Gaussianization scheme transforms any source model to a spherical Gaussian by alternating between linear ICA transformations and nonlinear histogram matching to map marginal densities to Gaussians [28]. However, in general, the overall transformation of iterative Gaussianization is an alternating concatenation of many linear/nonlinear transformations, and results in a substantial distortion of the original source space. For the special case of ESDs, RG provides a simple one-step procedure with minimal distortion. Another nonlinear transform that has also been shown to be able to reduce higher-order dependencies in natural signals is divisive normalization [15]. In the extended version of this paper [13], we show that there is no ESD source model for whose dependencies can be completely eliminated by divisive normalization. On the other hand, divisive normalization provides a rough approximation to RG, which suggests that RG might provide a more principled justification for normalization-like nonlinear behaviors seen in biological sensory systems.

There are a number of extensions of RG that are worth considering in the context of signal representation. First, we are interested in specific sub-families of ESD for which the nonlinear mapping of signal amplitudes in RG may be expressed in closed form. Second, the RG methodology provides a solution to the efficient coding problem for ESD signals in the noise-free case, and it is worthwhile to consider how the solution would be affected by the presence of sensor and/or channel noise. Third, we have shown that RG substantially reduces dependency for nearby samples of bandpass filtered image/sound, but that performance worsens as the coefficients become more separated, where their joint densities are closer to factorial. Recent models of natural images [29, 30] have used Markov random fields based on local elliptically symmetric models, and these are seen to provide a natural transition of pairwise joint densities from elliptically symmetric to factorial. We are currently exploring extensions of the RG methodology to such global models. And finally, we are currently examining the statistics of signals after local RG transformations, with the expectation that remaining statistical regularities (e.g., orientation and phase dependencies in images) can be studied, modeled and removed with additional transformations.

## References

[1]  F Attneave. Some informational aspects of visual perception. *Psych. Rev.*, 61:183–193, 1954.

---

[1]Similar results for the comparison of ICA to PCA were obtained with a slightly different method of removing the mean values of each block [8].

[2] H B Barlow. Possible principles underlying the transformation of sensory messages. In W A Rosenblith, editor, *Sensory Communication*, pages 217–234. MIT Press, Cambridge, MA, 1961.

[3] B A Olshausen and D J Field. Emergence of simple-cell receptive field properties by learning a sparse code for natural images. *Nature*, 381:607–609, 1996.

[4] A van der Schaaf and J H van Hateren. Modelling the power spectra of natural images: Statistics and information. *Vision Research*, 28(17):2759–2770, 1996.

[5] A J Bell and T J Sejnowski. The 'independent components' of natural scenes are edge filters. *Vision Research*, 37(23):3327–3338, 1997.

[6] M S Lewicki. Efficient coding of natural sounds. *Nature Neuroscience*, 5(4):356–363, 2002.

[7] R. Baddeley. Searching for filters with "interesting" output distributions: an uninteresting direction to explore. *Network*, 7:409–421, 1996.

[8] Matthias Bethge. Factorial coding of natural images: how effective are linear models in removing higher-order dependencies? *J. Opt. Soc. Am. A*, 23(6):1253–1268, 2006.

[9] B Wegmann and C Zetzsche. Statistical dependence between orientation filter outputs used in an human vision based image code. In *Proc Visual Comm. and Image Processing*, volume 1360, pages 909–922, Lausanne, Switzerland, 1990.

[10] E P Simoncelli. Statistical models for images: Compression, restoration and synthesis. In *Proc 31st Asilomar Conf on Signals, Systems and Computers*, volume 1, pages 673–678, Pacific Grove, CA, November 2-5 1997. IEEE Computer Society.

[11] M J Wainwright and E P Simoncelli. Scale mixtures of Gaussians and the statistics of natural images. In S. A. Solla, T. K. Leen, and K.-R. Müller, editors, *Adv. Neural Information Processing Systems (NIPS*99)*, volume 12, pages 855–861, Cambridge, MA, May 2000. MIT Press.

[12] K.T. Fang, S. Kotz, and K.W. Ng. *Symmetric Multivariate and Related Distributions*. Chapman and Hall, London, 1990.

[13] S. Lyu and E. P. Simoncelli. Nonlinear extraction of "independent components" of elliptically symmetric densities using radial Gaussianization. Technical Report TR2008-911, Computer Science Technical Report, Courant Inst. of Mathematical Sciences, New York University, April 2008.

[14] D. Nash and M. S. Klamkin. A spherical characterization of the normal distribution. *Journal of Multivariate Analysis*, 55:56–158, 1976.

[15] O Schwartz and E P Simoncelli. Natural signal statistics and sensory gain control. *Nature Neuroscience*, 4(8):819–825, August 2001.

[16] William Feller. *An Introduction to Probability Theory and Its Applications*, volume 1. Wiley, January 1968.

[17] C Zetzsche and G Krieger. The atoms of vision: Cartesian or polar? *J. Opt. Soc. Am. A*, 16(7), July 1999.

[18] J. Huang and D. Mumford. Statistics of natural images and models. In *IEEE International Conference on Computer Vision and Pattern Recognition (CVPR)*, 1999.

[19] A Srivastava, X Liu, and U Grenander. Universal analytical forms for modeling image probability. *IEEE Pat. Anal. Mach. Intell.*, 24(9):1200–1214, Sep 2002.

[20] Y. Teh, M. Welling, and S. Osindero. Energy-based models for sparse overcomplete representations. *Journal of Machine Learning Research*, 4:1235–1260, 2003.

[21] P I M Johannesma. The pre-response stimulus ensemble of neurons in the cochlear nucleus. In *Symposium on Hearing Theory (IPO)*, pages 58–69, Eindhoven, Holland, 1972.

[22] M. Studeny and J. Vejnarova. The multiinformation function as a tool for measuring stochastic dependence. In M. I. Jordan, editor, *Learning in Graphical Models*, pages 261–297. Dordrecht: Kluwer., 1998.

[23] T. Cover and J. Thomas. *Elements of Information Theory*. Wiley-Interscience, 2nd edition, 2006.

[24] A. Kraskov, H. Stögbauer, and P. Grassberger. Estimating mutual information. *Phys. Rev. E*, 69(6):66–82, Jun 2004.

[25] E. G. Learned-Miller and J. W. Fisher. ICA using spacings estimates of entropy. *Journal of Machine Learning Research*, 4(1):1271–1295, 2000.

[26] J H van Hateren and A van der Schaaf. Independent component filters of natural images compared with simple cells in primary visual cortex. *Proc. R. Soc. Lond. B*, 265:359–366, 1998.

[27] A. Hyvärinen. Fast and robust fixed-point algorithms for independent component analysis. *IEEE Transactions on Neural Networks*, 10(3):626–634, 1999.

[28] Scott Saobing Chen and Ramesh A. Gopinath. Gaussianization. In *Advances in Neural Computation Systems (NIPS)*, pages 423–429, 2000.

[29] S. Roth and M. Black. Fields of experts: A framework for learning image priors. In *IEEE Conference on Computer Vision and Patten Recognition (CVPR)*, volume 2, pages 860–867, 2005.

[30] S Lyu and E P Simoncelli. Statistical modeling of images with fields of Gaussian scale mixtures. In B Schölkopf, J Platt, and T Hofmann, editors, *Adv. Neural Information Processing Systems 19*, volume 19, Cambridge, MA, May 2007. MIT Press.

